# Analysis of Bit Error Probability of Direct-Sequence CDMA Multiuser Demodulators

**Toshiyuki Tanaka**
Department of Electronics and Information Engineering
Tokyo Metropolitan University
Hachioji, Tokyo 192-0397, Japan
*tanaka@eei.metro-u.ac.jp*

## Abstract

We analyze the bit error probability of multiuser demodulators for direct-sequence binary phase-shift-keying (DS/BPSK) CDMA channel with additive gaussian noise. The problem of multiuser demodulation is cast into the finite-temperature decoding problem, and replica analysis is applied to evaluate the performance of the resulting MPM (Marginal Posterior Mode) demodulators, which include the optimal demodulator and the MAP demodulator as special cases. An approximate implementation of demodulators is proposed using analog-valued Hopfield model as a naive mean-field approximation to the MPM demodulators, and its performance is also evaluated by the replica analysis. Results of the performance evaluation shows effectiveness of the optimal demodulator and the mean-field demodulator compared with the conventional one, especially in the cases of small information bit rate and low noise level.

## 1 Introduction

The CDMA (Code-Division-Multiple-Access) technique [1] is important as a fundamental technology of digital communications systems, such as cellular phones. The important applications include realization of spread-spectrum multipoint-to-point communications systems, in which multiple users share the same communication channel. In the multipoint-to-point system, each user modulates his/her own information bit sequence using a spreading code sequence before transmitting it, and the receiver uses the same spreading code sequence for demodulation to obtain the original information bit sequence. Different users use different spreading code sequences so that the demodulation procedure randomizes and thus suppresses multiple access interference effects of transmitted signal sequences sent from different users.

The direct-sequence binary phase-shift-keying (DS/BPSK) [1] is the basic method among various methods realizing CDMA, and a lot of studies have been done on it. Use of Hopfield-type recurrent neural network has been proposed as an implementation of a multiuser demodulator [2]. In this paper, we analyze the bit error probability of the neural multiuser demodulator applied to demodulation of DS/BPSK CDMA channel.

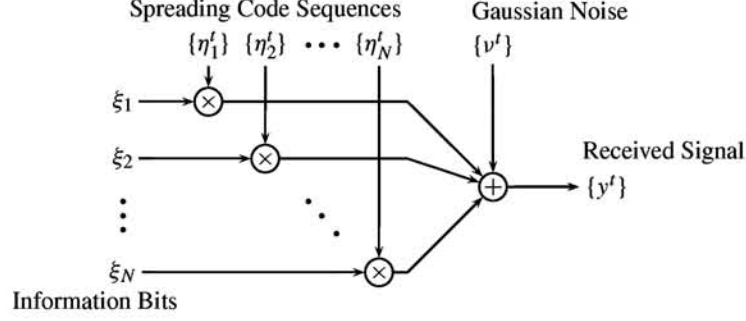

Figure 1: DS/BPSK CDMA model

## 2 DS/BPSK CDMA system

We assume that a single Gaussian channel is shared by $N$ users, each of which wishes to transmit his/her own information bit sequence. We also take a simplifying assumption, that all the users are completely synchronized with each other, with respect not only to the chip timing but also to the information bit timing. We focus on any of the time intervals corresponding to the duration of one information bit. Let $\xi_i \in \{-1, 1\}$ be the information bit to be transmitted by user $i$ ($i = 1, \ldots, N$) during the time interval, and $P$ be the number of the spreading code chips (clocks) per information bit. For simplicity, the spreading code sequences for the users are assumed to be random bit sequences $\{\eta_i^t; t = 1, \ldots, P\}$, where $\eta_i^t$'s are independent and identically distributed (i.i.d.) binary random variables following $\text{Prob}[\eta_i^t = \pm 1] = 1/2$.

User $i$ modulates the information bit $\xi_i$ by the spreading code sequence and transmits the modulated sequence $\{\xi_i \eta_i^t; t = 1, \ldots, P\}$ (with carrier modulation, in actual situations). Assuming that power control [3] is done perfectly so that every transmitted sequences arrive at the receiver with the same intensity, the received signal sequence (after baseband demodulation) is $\{y^t; t = 1, \ldots, P\}$, with

$$y^t = \sum_{i=1}^{N} \eta_i^t \xi_i + \nu^t, \qquad (1)$$

where $\nu^t \sim N(0, \sigma_s^2)$ is i.i.d. gaussian noise. This system is illustrated in Fig. 1.

At the receiver side, one has to estimate the information bits $\{\xi_i\}$ based on the knowledge of the received signal $\{y^t\}$ and the spreading code sequences $\{\eta_i^t\}$ for the users. The demodulator refers to the system which does this task. Accuracy of the estimation depends on what demodulator one uses. Some demodulators are introduced in Sect. 3, and analytical results for their performance is derived in Sect. 4.

## 3 Demodulators

### 3.1 Conventional demodulator

The conventional demodulator (CD) [1–3] estimates the information bit $\xi_i$ using the spreading code sequence $\{\eta_i^t; \ t = 1, \ldots, P\}$ for the user $i$, by

$$\hat{\xi}_i^{(CD)} = \mathrm{sgn}(h_i), \quad h_i \equiv \frac{1}{N} \sum_{t=1}^{P} \eta_i^t y^t. \tag{2}$$

We can rewrite $h_i$ as

$$h_i = \frac{P}{N}\xi_i + \frac{1}{N} \sum_{t=1}^{P} \sum_{k \neq i}^{N} \eta_i^t \eta_k^t \xi_k + \frac{1}{N} \sum_{t=1}^{P} \eta_i^t \nu^t. \tag{3}$$

The second and third terms of the right-hand side represent the effects of multiple access interference and noise, respectively. CD would give the correct information bit in the single-user ($N = 1$), and no noise ($\nu^t \equiv 0$) case, but estimation may contain some errors in the multiple-user and/or noisy cases.

### 3.2 MAP demodulator

The accuracy of the estimation would be significantly improved if the demodulator knows the spreading code sequences for *all* $N$ users and makes full use of them by simultaneously estimating the information bits for all the users (the *multiuser demodulator*). This is the case, for example, for a base station receiving signals from many users. A common approach to the multiuser demodulation is to use the MAP decoding, which estimates the information bits $\{s_i = \hat{\xi}_i\}$ by maximizing the posterior probability $p(\{\xi_i\}|\{y^t\})$. We call this kind of multiuser demodulator the MAP demodulator[1].

When we assume uniform prior for the information bits, the posterior probability is explicitly given by

$$p(s|\{y^t\}) = Z^{-1} \exp\!\left(-\beta_s H(s)\right), \tag{4}$$

where

$$H(s) \equiv \frac{1}{2} s^T W s - h^T s, \tag{5}$$

$\beta_s \equiv N/\sigma_s^2$, $s \equiv (s_i)$, $h \equiv (h_i)$, and $W \equiv (w_{ij})$ is the sample covariance of the spreading code sequences,

$$w_{ij} = \frac{1}{N} \sum_{t=1}^{P} \eta_i^t \eta_j^t. \tag{6}$$

The problem of MAP demodulation thus reduces to the following minimization problem:

$$\hat{\xi}^{(MAP)} = \arg \min_{s \in \{-1, 1\}^N} H(s). \tag{7}$$

### 3.3 MPM demodulator

Although the MAP demodulator is sometimes referred to as "optimal," actually it is not so in terms of the common measure of performance, i.e., the bit error probability $P_b$, which is

related to the overlap $M \equiv (1/N) \sum_{i=1}^{N} \xi_i \hat{\xi}_i$ between the original information bits $\{\xi_i\}$ and their estimates $\{\hat{\xi}_i\}$ as

$$P_b = \frac{1 - M}{2}. \tag{8}$$

The 'MPM (Marginal Posterior Mode [4]) demodulator,' with the inverse temperature $\beta$, is defined as follows:

$$\hat{\xi}_i^{(\mathrm{MPM})} = \mathrm{sgn}(\langle s_i \rangle_\beta), \tag{9}$$

where $\langle \cdot \rangle_\beta$ refers to the average with respect to the distribution

$$p_\beta(s) = Z(\beta)^{-1} \exp(-\beta H(s)). \tag{10}$$

Then, we can show that the MPM demodulator with $\beta = \beta_s$ is the optimal one minimizing the bit error probability $P_b$. It is a direct consequence of general argument on optimal decoders [5]. Note that the MAP demodulator corresponds to the MPM demodulator in the $\beta \to +\infty$ limit (the *zero-temperature* demodulator).

## 4 Analysis

### 4.1 Conventional demodulator

In the cases where we can assume that $N$ and $P$ are both large while $\alpha \equiv P/N = O(1)$, evaluation of the overlap $M$, and therefore the bit error probability $P_b$, for those demodulators are possible. For CD, simple application of the central limit theorem yields

$$M = \mathrm{erf}\left(\sqrt{\frac{\alpha}{2(1 + 1/\beta_s)}}\right), \tag{11}$$

where

$$\mathrm{erf}(x) \equiv \frac{2}{\sqrt{\pi}} \int_0^x e^{-t^2} dt \tag{12}$$

is the error function.

### 4.2 MPM demodulator

For the MPM demodulator with inverse temperature $\beta$, we have used the replica analysis to evaluate the bit error probability $P_b$. Assuming that $N$ and $P$ are both large while $\alpha \equiv P/N = O(1)$, and that the macroscopic properties of the demodulator are self-averaging with respect to the randomness of the information bits, of the spreading codes, and of the noise, we evaluate the quenched average of the free energy $\langle\langle \log Z \rangle\rangle$ in the thermodynamic limit $N \to \infty$, where $\langle\langle \cdot \rangle\rangle$ denotes averaging over the information bits and the noise.

Evaluation of the overlap $M$ (within replica-symmetric (RS) ansatz) requires solving saddle-point problem for scalar variables $\{m, q, E, F\}$. The saddle-point equations are

$$m = \int Dz \, \tanh(\sqrt{F}z + E), \quad q = \int Dz \, \tanh^2(\sqrt{F}z + E)$$
$$E = \frac{\alpha\beta}{1 + \beta(1 - q)}, \qquad F = \frac{\alpha\beta^2}{[1 + \beta(1 - q)]^2}\left[1 - 2m + q + \frac{1}{\beta_s}\right] \tag{13}$$

where $Dz \equiv (1/\sqrt{2\pi})e^{-z^2/2}dz$ is the gaussian measure. The overlap $M$ is then given by

$$M = \int Dz \, \mathrm{sgn}(\sqrt{F}z + E), \tag{14}$$

from which $P_b$ is evaluated via (8). This is the first main result of this paper.

### 4.3 MAP demodulator: Zero-temperature limit

Taking the zero-temperature limit $\beta \to +\infty$ of the result for the MPM demodulator yields the result for the MAP demodulator. Assuming that $q \to 1$ as $\beta \to +\infty$, while $\beta(1-q)$ remains finite in this limit, the saddle-point equations reduce to

$$M = m = \text{erf}\left(\sqrt{\frac{\alpha}{2(2 - 2m + 1/\beta_s)}}\right). \tag{15}$$

It is found numerically, however, that the assumption $q \to 1$ is not valid for small $\alpha$, so that we have to solve the original saddle-point equations in such cases.

### 4.4 Optimal demodulator: The case $\beta = \beta_s$

Letting $\beta = \beta_s$ in the result for the MPM demodulator gives the optimal demodulator minimizing the bit error probability. In this case, it can be shown that $m = q$ and $E = F$ hold for the solutions of the saddle-point equations (13).

### 4.5 Demodulator using naive mean-field approximation

Since solving the MAP or MPM demodulation problem is in general NP complete, we have to consider approximate implementations of those demodulators which are sub-optimal. A straightforward choice is the mean-field approximation (MFA) demodulator, which uses the analog-valued Hopfield model as the naive mean-field approximation to the finite-temperature demodulation problem[2]. The solution $\{m_i\}$ of the mean-field equations

$$m_i = \tanh\left[\beta\left(-\sum_j w_{ij} m_j + h_i\right)\right] \tag{16}$$

gives an approximation to $\{\langle s_i \rangle_\beta\}$, from which we have the mean-field approximation to the MPM estimates, as

$$\hat{\xi}_i^{(\text{MFA})} = \text{sgn}(m_i). \tag{17}$$

The macroscopic properties of the MFA demodulator can be derived by the replica analysis as well, along the line proposed by Bray et al. [6] We have derived the following saddle-point equations:

$$m = \int Dz\, f(z), \quad \bar{\chi} = \frac{1}{\sqrt{F}} \int Dz\, z f(z), \quad q = \int Dz\, [f(z)]^2$$
$$E = \frac{\alpha\beta}{1 + \beta\bar{\chi}}, \qquad F = \frac{\alpha\beta^2}{[1 + \beta\bar{\chi}]^2}\left[1 - 2m + q + \frac{1}{\beta_s}\right], \tag{18}$$

where $f(z)$ is the function defined by

$$f(z) = \tanh\left[\sqrt{F}z - Ef(z) + E\right]. \tag{19}$$

$f(z)$ is a single-valued function of $z$ since $E$ is positive. The overlap $M$ is then calculated by

$$M = \int Dz\, \text{sgn}(f(z)). \tag{20}$$

This is the second main result of this paper.

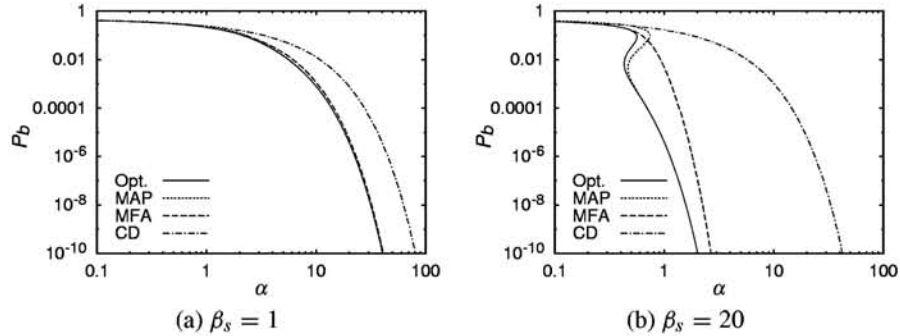

Figure 2: Bit error probability for various demodulators.

### 4.6 AT instability

The AT instability [7] refers to the bifurcation of a saddle-point solution without replica symmetry from the replica-symmetric one. In this paper we follow the usual convention and assume that the first such destabilization occurs in the so-called "replicon mode [8]." As the stability condition of the RS saddle-point solution for the MPM demodulator, we obtain

$$\alpha - E^2 \int Dz \ \text{sech}^4(\sqrt{F}z + E) = 0. \tag{21}$$

For the MFA demodulator, we have

$$\alpha - E^2 \int Dz \left[ \frac{1 - f(z)^2}{1 + E(1 - f(z)^2)} \right]^2 = 0. \tag{22}$$

The RS solution is stable as long as the left-hand side of (21) or (22) is positive.

## 5 Performance evaluation

The saddle-point equations (13) and (18) can be solved numerically to evaluate the bit error probability $P_b$ of the MPM demodulator and its naive mean-field approximation, respectively. We have investigated four demodulators: the optimal one ($\beta = \beta_s$), MAP, MFA (with $\beta = \beta_s$, i.e., the naive mean-field approximation to the optimal one), and CD. The results are summarized in Fig. 2 (a) and (b) for two cases with $\beta_s = 1$ and 20, respectively.

Increasing $\alpha$ corresponds to relatively lowering the information bit rate, so that $P_b$ should become small as $\alpha$ gets larger, which is in consistent with the general trend observed in Fig. 2. The optimal demodulator shows consistently better performance than CD, as expected. The MAP demodulator marks almost the same performance as the optimal one (indeed the result of the MAP demodulator is nearly the same as that of the optimal demodulator in the case $\beta_s = 1$, so they are indistinguishable from each other in Fig. 2 (a)).

We also found that the performance of the optimal, MAP, and MFA demodulators is significantly improved in the large-$\alpha$ region when the variance $\sigma_s^2$ of the noise is small relative to $N$, the number of the users. For example, in order to achieve practical level of bit error probability, $P_b \sim 10^{-5}$ say, in the $\beta_s = 1$ case the optimal and MAP demodulators allow information bit rate 2 times faster than CD does. On the other hand, in the $\beta_s = 20$ case they allow information bit rate as much as 20 times faster than CD, which demonstrates that significant process gain is achieved by the optimal and MAP demodulators in such cases.

The MFA demodulator with $\beta = \beta_s$ showed the performance competitive with the optimal one for the $\beta_s = 1$ case. Although the MFA demodulator fell behind the optimal and MAP demodulators in the performance for the $\beta_s = 20$ case, it still had process gain which allows about 10 times faster information bit rate than CD does. Moreover, we observed, using (22), that the RS saddle-point solution for the MFA demodulator with $\beta = \beta_s$ was stable with respect to replica symmetry breaking (RSB), and thus RS ansatz was indeed valid for the MFA solution. It suggests that the free energy landscape is rather simple for these cases, making it easier for the MFA demodulator to find a good solution. This argument provides an explanation as to why finite-temperature analog-valued Hopfield models, proposed heuristically by Kechriotis and Manolakos [2], exhibited better performance in their numerical experiments. We also found that the RS saddle-point solution for the optimal demodulator was stable with respect to RSB over the whole range investigated, whereas the solution for the MAP demodulator was found to be unstable. This observation suggests the possibility to construct efficient near-optimal demodulators using advanced mean-field approximations, such as the TAP approach [9, 10].

### Acknowledgments

This work is supported in part by Grant-in-Aid for Scientific Research from the Ministry of Education, Science, Sports and Culture, Japan.

## Footnotes

[1]The MAP demodulator refers to the same one as what is frequently called the "maximum-likelihood (ML) demodulator" in the literature.

[2]The proposal by Kechriotis and Manolakos [2] is to use the Hopfield model for an approximation to the *MAP* demodulation. The proposal in this paper goes beyond theirs in that the analog-valued Hopfield model is used to approximate not the MAP demodulator in the zero-temperature limit but the MPM demodulators directly, including the optimal one.

### References

[1] M. K. Simon, J. K. Omura, R. A. Scholtz, and B. K. Levitt, *Spread Spectrum Communications Handbook*, Revised Ed., McGraw-Hill, 1994.

[2] G. I. Kechriotis and E. S. Manolakos, "Hopfield neural network implementation of the optimal CDMA multiuser detector," *IEEE Trans. Neural Networks*, vol. 7, no. 1, pp. 131–141, Jan. 1996.

[3] A. J. Viterbi, *CDMA: Principles of Spread Spectrum Communication*, Addison-Wesley, Reading, Massachusetts, 1995.

[4] G. Winkler, *Image Analysis, Random Fields and Dynamic Monte Carlo Methods*, Springer-Verlag, Berlin, Heidelberg, 1995.

[5] Y. Iba, "The Nishimori line and Bayesian statistics," *J. Phys. A: Math. Gen.*, vol. 32, no. 21, pp. 3875–3888, May 1999.

[6] A. J. Bray, H. Sompolinsky, and C. Yu, "On the 'naive' mean-field equations for spin glasses," *J. Phys. C: Solid State Phys.*, vol. 19, no. 32, pp. 6389–6406, Nov. 1986.

[7] J. R. L. de Almeida and D. J. Thouless, "Stability of the Sherrington-Kirkpatrick solution of a spin glass mode," *J. Phys. A: Math. Gen.*, vol. 11, no. 5, pp. 983–990, 1978.

[8] K. H. Fischer and J. A. Hertz *Spin Glasses*, Cambridge University Press, Cambridge, 1991.

[9] D. J. Thouless, P. W. Anderson, and R. G. Palmer, "Solution of 'Solvable model of a spin glass'," *Phil. Mag.*, vol. 35, no. 3, pp. 593–601, 1977.

[10] Y. Kabashima and D. Saad, "The belief in TAP," in M. S. Kearns et al.(eds.), *Advances in Neural Information Processing Systems*, vol. 11, The MIT Press, pp. 246–252, 1999.
